# SPREADING ACTIVATION OVER DISTRIBUTED MICROFEATURES

James Hendler[*]
Department of Computer Science
University of Maryland
College Park, MD 20742

## ABSTRACT

One attempt at explaining human inferencing is that of spreading activation, particularly in the structured connectionist paradigm. This has resulted in the building of systems with semantically nameable nodes which perform inferencing by examining the patterns of activation spread. In this paper we demonstrate that simple structured network inferencing can be performed by passing activation over the weights learned by a distributed algorithm. Thus, an account is provided which explains a well-behaved relationship between structured and distributed connectionist approaches.

## INTRODUCTION

A primary difference between the neural networks of 20 years ago and the current generation of connectionist models is the addition of mechanisms which permit the system to create an internal representation. These subsymbolic, semantically unnameable, features which are induced by connectionist learning algorithms have been discussed as being of import both in structured and distributed connectionist networks (cf. Feldman and Ballard, 1982; Rumelhart and McClelland, 1986). The fact that network learning algorithms can create these *microfeatures* is not, however. enough in itself to account for how cognition works. Most of what we call intelligent thought derives from being able to reason about the relations between objects, to hypothesize about events and things, etc. If we are to do cognitive modeling we must complete the story by explaining how networks can reason in the way that humans (or other intelligent beings) do.

One attempt at explaining such reasoning is that of spreading activation in the structured connectionist and marker-passing (cf. Charniak, 1983; Hendler, 1987)

---

[*] The author is also affiliated with the Institute for Advanced Computer Studies and the Systems Research Center at the University of Maryland. Funding for this work was provided in part by Office of Naval Research Grant N00014-88-K-0560.

approaches. In these systems semantically nameable nodes permit an energy spread, and reasoning about the world is accounted for by looking at either stable configurations of the activation (the structured connectionist approach) or at the paths found by examining intersections among the nodes (the marker-passing technique). In this paper we will demonstrate that simple structured-network-like inferencing can be performed by passing activation over the weights learned by a distributed algorithm. Thus, an account is provided which explains a well-behaved relationship between structured and distributed connectionist approaches.

## THE SPREADING ACTIVATION MODEL

In this paper we will demonstrate that local connectionist-like networks can be built by spreading activation over the microfeatures learned by a distributed network. To show this, we start with a simple example which demonstrates the activation spreading mechanism used. The particular network we will use in this example is a 6-3-8 three-layer network trained by the back-propagation learning algorithm. The training set used is shown in table 1. The weights between the output nodes and hidden units which are learned by the network (after learning to the 90% level for a typical run) are shown in figure 1.

TABLE 1. Training Set for Example 1.

| Input Pattern | Output Pattern |
|---|---|
| 0 0 0 0 0 0 | 1 0 0 0 0 0 0 0 |
| 0 0 0 0 1 1 | 0 1 0 0 0 0 0 0 |
| 0 0 1 1 0 0 | 0 0 1 0 0 0 0 0 |
| 0 0 1 1 1 1 | 0 0 0 1 0 0 0 0 |
| 1 1 0 0 0 0 | 0 0 0 0 1 0 0 0 |
| 1 1 0 0 1 1 | 0 0 0 0 0 1 0 0 |
| 1 1 1 1 0 0 | 0 0 0 0 0 0 1 0 |
| 1 1 1 1 1 1 | 0 0 0 0 0 0 0 1 |

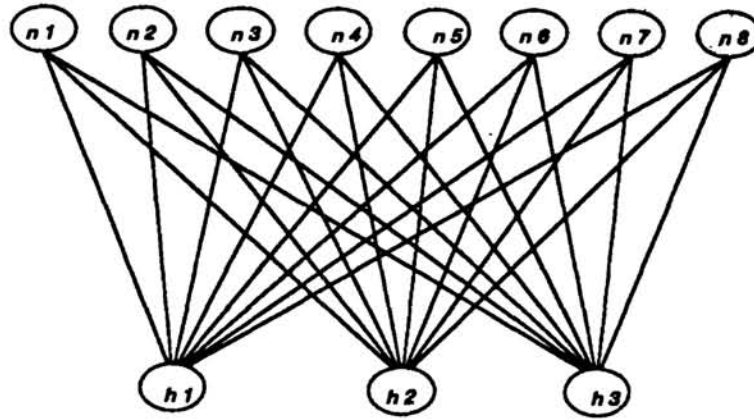

**Weights**

|     | h1     | h2     | h3     |
|-----|--------|--------|--------|
| n1  | -4.98  | 4.40   | -2.82  |
| n2  | -6.99  | -4.99  | -2.23  |
| n3  | -6.11  | 3.49   | 0.30   |
| n4  | -6.37  | -4.68  | 2.53   |
| n5  | 4.36   | 3.73   | -5.09  |
| n6  | 4.38   | -5.97  | -3.67  |
| n7  | 0.89   | 1.07   | 3.32   |
| n8  | 3.88   | -6.95  | 1.88   |

Figure 1. Weights Learned by Back Propagation

To understand how the activation spreads, let us examine what occurs when activation is started at node *n1* with a weight of *1*. This activation strength is divided by the outbranching of the node and then multiplied by the weight of each link to the hidden units. Thus activation flows from *n1* to *h1* with a strength of *1/3 x Weight(n1,h1)*. A similar computation is made to each of the other hidden units. This activation now spreads to each of the other output nodes in turn. Thus, *n2* would gain activation of

$$Activation(h1) \; x \; Weight(n2,h1)/8 \; +$$
$$Activation(h2) \; x \; Weight(n2,h2)/8 \; +$$
$$Activation(h3) \; x \; Weight(n2,h3)/8$$

or *.80* from *n1*.

Table 2 shows a graph of the activation spread between the output units. The table, which is symmetric, can thus be read as showing the output at each of the other units when an activation strength of *1* is placed at the named node. Looking at the table we see that the highest activation occurs among nodes which share the most features of the input (i.e. same value and position) while the lowest is seen among those patterns sharing the fewest features.

However, as well as having this property, table 2 can be seen as providing a matrix which specifies the weights between the output nodes if viewed as a structured network. That is, $n1$ is connected to $n2$ by a strength of $+.80$, to $n3$ by a strength of $+1.03$, etc. Thus, by using this technique distributed representations can be turned into connectivity weights for structured networks. When non–orthogonal weights are used, the same activation–spreading algorithm produces a structured network which can be used for more complex inferencing than can the distributed network alone.

We demonstrate this by a simple, and again contrived, example. This example is motivated by Gary Cottrell's structured model for word sense disambiguation (Cottrell, 1985). Cottrell, using weights derived by hand, demonstrated that a structured connectionist network could distinguish both word–sense and case–slot assignments for ambiguous lexical items. Presented with the sentence "John threw the fight" the system would activate a node for one meaning of "throw," presented with "John threw the ball" it would come up with another. The nodes of Cottrell's network included words (John, Threw, etc.), word senses (John1, Propel, etc.) and case–slots (TAGT (agent of the throw), PAGT (agent of the Propel), etc.).

TABLE 2. Activation Spread in 6–3–8 Network.

|      | $n1$   | $n2$   | $n3$   | $n4$   | $n5$   | $n6$   | $n7$   | $n8$   |
|------|--------|--------|--------|--------|--------|--------|--------|--------|
| $n1$ | •      | .80    | 1.03   | .17    | .38    | −1.57  | −.38   | −2.3   |
| $n2$ | .80    | •      | 1.02   | 2.6    | −1.57  | .31    | −.79   | .14    |
| $n3$ | 1.03   | 1.02   | •      | .97    | −.63   | −2.03  | −.03   | −1.97  |
| $n4$ | .17    | 2.60   | .97    | •      | −2.42  | −.38   | −.09   | .52    |
| $n5$ | .38    | −1.57  | −.63   | −2.42  | •      | .64    | −.38   | −.77   |
| $n6$ | −1.57  | .31    | −2.03  | −.38   | .64    | •      | −.6    | 2.14   |
| $n7$ | −.38   | −.79   | −.03   | −.09   | −.38   | −.6    | •      | .09    |
| $n8$ | −2.3   | −.14   | −1.97  | .52    | −.77   | 2.14   | .09    | •      |

To duplicate Gary's network via training, we presented a 3–layer backprop network with a training set in which distributed patterns, very loosely corresponding to a "dictionary" of word encodings[1] were associated with a vector representing each of the individual nodes which would be represented in Cottrell's system, but with no structure. Thus, each element in the training set is

---

1– Which in any realistic system would some day be replaced by actual signal processing outputs or other representations of actual word pronunciation forms.

a 16 bit vector (representing a four word sentence, each word as a 4 bit pattern), associated with another 16 bit vector representing the nodes

Bob1 John1 propel threw fight1 ball1 pagt pobj tagt tobj bob john threw the fight ball

For this example, the system was trained on the encodings of the four sentences

John threw the ball
John threw the fight
Bob threw the ball
Bob threw the fight

with the output set high for those objects in the second vector which were appropriately associated. as shown in Table 3.

TABLE 3. Training Set for Example 2.

| Input Pattern | Output Pattern |
|---|---|
| 0110 0001 0101 0010 | 1001100011101110 |
| 0110 0001 0101 1010 | 1010011100101101 |
| 1001 0001 0101 0010 | 0101100011011110 |
| 1001 0001 0101 1010 | 0110011100011101 |

Upon completion of the learning, the activation spreading algorithm was used to derive a table of connectivity weights between the output units as shown in table 4.

These weights were then transferred into a local connectionist simulator and a very simple activation spreading model was used to examine the results. When we run the simulator. using the activation spreading over learned weights. exactly the results produced by Cottrell's network are seen. Thus:

Activation from the nodes corresponding to *john. throw, the.* and *fight* cause a positive activation at the node for "Throw" and a negative activation at the node for "Propel."
while
Activation from *john throw the ball* spread positively to "Propel" and not to "throw."

Further, other effects which are also predicted by Cottrell's model are seen:

Activation at *TAGT* and *TOBJ* spreads positive activation to *Throw* and not to *Propel.*
and
Activation at *PAGT* and *POBJ* causes a spread to *Propel* but not to *Throw.*

TABLE 4. Connectivity Weights for Example 2.

```
***  -0.12  0.01 -0.01 -0.01  0.01  0.01  0.01 -0.01 -0.01  0.12 -0.12 -0.03 -0.03 -0.01  0.00
-0.12  ***  -0.01  0.01  0.01 -0.01 -0.01 -0.01  0.01  0.01 -0.12  0.12  0.03  0.03  0.01 -0.01
 0.01 -0.01  ***  -0.04 -0.04  0.04  0.04  0.05 -0.05 -0.04  0.01 -0.01 -0.02 -0.02 -0.04  0.04
-0.01  0.01 -0.04  ***   0.04 -0.04 -0.04 -0.05  0.05  0.04 -0.01  0.01  0.02  0.02  0.04 -0.04
-0.01  0.01 -0.04  0.04  ***  -0.04 -0.05 -0.05  0.05  0.04 -0.01  0.01  0.02  0.02  0.04 -0.04
 0.01 -0.01  0.04 -0.04 -0.04  ***   0.05  0.05 -0.05 -0.04  0.01 -0.01 -0.02 -0.02 -0.04  0.04
 0.01 -0.01  0.04 -0.04 -0.05  0.05  ***   0.05 -0.05 -0.05  0.01 -0.00 -0.02 -0.02 -0.04  0.05
 0.01 -0.01  0.05 -0.05 -0.05  0.05  0.05  ***  -0.05 -0.05  0.01 -0.01 -0.02 -0.02 -0.04  0.05
-0.01  0.01 -0.05  0.05  0.05 -0.05 -0.05 -0.05  ***   0.05 -0.01  0.01  0.02  0.03  0.04 -0.05
-0.01  0.01 -0.04  0.04  0.04 -0.04 -0.05 -0.05  0.05  ***  -0.01  0.01  0.02  0.02  0.04 -0.05
 0.12 -0.12  0.01 -0.01 -0.01  0.01  0.01  0.01 -0.01 -0.01  ***  -0.12 -0.03 -0.03 -0.01  0.01
-0.12  0.12 -0.01  0.01  0.01 -0.01 -0.00 -0.01  0.01  0.01 -0.12  ***   0.03  0.03  0.01 -0.00
-0.03  0.03 -0.02  0.02  0.02 -0.02 -0.02 -0.02  0.02  0.02 -0.03  0.03  ***   0.20  0.02 -0.02
-0.03  0.03 -0.02  0.02  0.02 -0.02 -0.02 -0.02  0.03  0.02  0.03  0.03  0.20  ***   0.02 -0.02
-0.01  0.01 -0.04  0.04  0.04 -0.04 -0.04 -0.04  0.04  0.04 -0.01  0.01  0.02  0.02  ***   0.04
 0.00 -0.01  0.04 -0.04 -0.04  0.04  0.05  0.05  0.05  0.05  0.01 -0.00  0.02  0.02 -0.04  ***
```

We believe that results like this one may argue that structured networks are integrally linked to distributed networks in that distributed network learning techniques may provide a fundamental basis for explaining the cognitive development of structured networks. In addition, we see that simple inferential reasoning can be produced using purely connectionist models.

## CONCLUDING REMARKS

We have attempted to show that a model using an activation spreading variant can be used to take learned connectionist models and perform some limited forms of inferencing upon them. Further, we have argued that this technique may provide a computational model in which structured networks can be learned and that structured networks provide the inferencing capabilities missing in purely distributed models. However, before we can truly further this claim, significant work remains to be done. We must extend and explore such models, particularly examining whether these types of techniques can be extended to handle the complexity that can be found in real-world problems and serious cognitive models.

In particular we are beginning an examination of two crucial issues: First, will the technique described above work for realistic problems? In particular, can the inferencing be designed to impact on the recognition by the distributed network? If so, one could see, for example, a speech recognition program coupled to a system like Cottrell's natural language system, providing a handle for a text understanding system. Similarly such a technique might allow the integration of top-down and bottom-up processing for vision and other such signal processing tasks.

Secondly, we wish to see if more complex spreading activation models could be hooked to this type of model. Could networks such as those proposed by Shastri (1985), Diederich (1985), and Pollack and Waltz (1982) which provide complex inferencing but require more structure than simply weights between units, be abstracted out of the learned weights? Two particular areas currently being pursued by the author, for example, focus on active inhibition models for determining whether portions of the network can be suppressed to provide more complex inferencing and the learning of structures given temporally ordered information.

## References

Charniak, E. Passing markers: A theory of contextual influence in language comprehension *Cognitive Science, 7(3)*, 1983. *171-190*.

Cottrell, G.W. *A Connectionist Approach to Word-Sense Disambiguation*, Doctoral Dissertation, Computer Science Dept., University of Rochester, 1985.

Diederich, J. *Parallelverarbeitung in netzwerk-basierten Systemen* PhD Dissertation, Dept. of Linguistics, University of Bielefeld, 1985.

Feldman, J.A. and Ballard, D.H. (1982). Connectionist models and their properties. *Cognitive Science, 6. 205-254*.

Hendler, J.A. *Integrating Marker-passing and Problem Solving: A spreading activation approach to improved choice in planning* Lawrence Erlbaum Associates, N.J., November, 1987.

Pollack, J.B. and Waltz, D.L Natural Language Processing using spreading activation and lateral inhibition *Proceedings of the Fourth International Conference of the Cognitive Science Society*, 1982, *50-53*.

Rumelhart, D.E. and McClelland, J.L. (eds.) *Parallel Distributed Computing* Cambridge, Ma.: MIT Press.

Shastri, L. *Evidential Reasoning in Semantic Networks: A formal theory and its parallel implementation* Doctoral Dissertation, Computer Science Department, University of Rochester, Sept., 1985.